# Learning continuous distributions:
# Simulations with field theoretic priors

**Ilya Nemenman**[1,2] **and William Bialek**[2]
[1] Department of Physics, Princeton University, Princeton, New Jersey 08544
[2] NEC Research Institute, 4 Independence Way, Princeton, New Jersey 08540
*nemenman@research.nj.nec.com, bialek@research.nj.nec.com*

## Abstract

Learning of a smooth but nonparametric probability density can be regularized using methods of Quantum Field Theory. We implement a field theoretic prior numerically, test its efficacy, and show that the free parameter of the theory ('smoothness scale') can be determined self consistently by the data; this forms an infinite dimensional generalization of the MDL principle. Finally, we study the implications of one's choice of the prior and the parameterization and conclude that the smoothness scale determination makes density estimation very weakly sensitive to the choice of the prior, and that even wrong choices can be advantageous for small data sets.

One of the central problems in learning is to balance 'goodness of fit' criteria against the complexity of models. An important development in the Bayesian approach was thus the realization that there does not need to be any extra penalty for model complexity: if we compute the total probability that data are generated by a model, there is a factor from the volume in parameter space—the 'Occam factor'—that discriminates against models with more parameters [1, 2]. This works remarkably well for systems with a finite number of parameters and creates a complexity 'razor' (after 'Occam's razor') that is almost equivalent to the celebrated Minimal Description Length (MDL) principle [3]. In addition, if the a priori distributions involved are strictly Gaussian, the ideas have also been proven to apply to some infinite–dimensional (nonparametric) problems [4]. It is not clear, however, what happens if we leave the finite dimensional setting to consider nonparametric problems which are not Gaussian, such as the estimation of a smooth probability density. A possible route to progress on the nonparametric problem was opened by noticing [5] that a Bayesian prior for density estimation is equivalent to a quantum field theory (QFT). In particular, there are field theoretic methods for computing the infinite dimensional analog of the Occam factor, at least asymptotically for large numbers of examples. These observations have led to a number of papers [6, 7, 8, 9] exploring alternative formulations and their implications for the speed of learning. Here we return to the original formulation of Ref. [5] and use numerical methods to address some of the questions left open by the analytic work [10]: What is the result of balancing the infinite dimensional Occam factor against the goodness of fit? Is the QFT inference optimal in using all of the information relevant for learning [11]? What happens if our learning problem is strongly atypical of the prior distribution?

Following Ref. [5], if $N$ i. i. d. samples $\{x_i\}, i = 1 \ldots N$, are observed, then the probability

that a particular density $Q(x)$ gave rise to these data is given by

$$P[Q(x)|\{x_i\}] = \frac{P[Q(x)]\prod_{i=1}^{N}Q(x_i)}{\int[dQ(x)]P[Q(x)]\prod_{i=1}^{N}Q(x_i)}, \tag{1}$$

where $P[Q(x)]$ encodes our a priori expectations of $Q$. Specifying this prior on a space of functions defines a QFT, and the optimal least square estimator is then

$$Q_{\text{est}}(x|\{x_i\}) = \frac{\langle Q(x)Q(x_1)Q(x_2)\ldots Q(x_N)\rangle^{(0)}}{\langle Q(x_1)Q(x_2)\ldots Q(x_N)\rangle^{(0)}}, \tag{2}$$

where $\langle\ldots\rangle^{(0)}$ means averaging with respect to the prior. Since $Q(x) \geq 0$, it is convenient to define an unconstrained field $\phi(x)$, $Q(x) \equiv (1/\ell_0)\exp[-\phi(x)]$. Other definitions are also possible [6], but we think that most of our results do not depend on this choice.

The next step is to select a prior that regularizes the infinite number of degrees of freedom and allows learning. We want the prior $\mathcal{P}[\phi]$ to make sense as a continuous theory, independent of discretization of $x$ on small scales. We also require that when we estimate the distribution $Q(x)$ the answer must be everywhere finite. These conditions imply that our field theory must be convergent at small length scales. For $x$ in one dimension, a minimal choice is

$$\mathcal{P}[\phi(x)] = \frac{1}{\mathcal{Z}}\exp\left[-\frac{\ell^{2\eta-1}}{2}\int dx \left(\frac{\partial^\eta\phi}{\partial x^\eta}\right)^2\right]\delta\left[\frac{1}{\ell_0}\int dx\,\mathrm{e}^{-\phi(x)} - 1\right], \tag{3}$$

where $\eta > 1/2$, $\mathcal{Z}$ is the normalization constant, and the $\delta$-function enforces normalization of $Q$. We refer to $\ell$ and $\eta$ as the *smoothness scale* and the *exponent*, respectively.

In [5] this theory was solved for large $N$ and $\eta = 1$:

$$\langle\prod_{i=1}^{N}Q(x_i)\rangle^{(0)} \approx \frac{1}{\ell_0^N}\exp\left(-S_{\text{eff}}[\phi_{\text{cl}}(x); \{x_i\}]\right), \tag{4}$$

$$S_{\text{eff}} = \int dx\left[\frac{\ell}{2}(\partial_x\phi_{\text{cl}})^2 + \frac{1}{2}\sqrt{\frac{N\mathrm{e}^{-\phi_{\text{cl}}}}{\ell\ell_0}}\right] + \sum_{j=1}^{N}\phi_{\text{cl}}(x_j), \tag{5}$$

$$\ell\partial_x^2\phi_{\text{cl}}(x) + \frac{N}{\ell_0}\mathrm{e}^{-\phi_{\text{cl}}(x)} = \sum_{j=1}^{N}\delta(x - x_j), \tag{6}$$

where $\phi_{\text{cl}}$ is the 'classical' (maximum likelihood, saddle point) solution. In the effective action [Eq. (5)], it is the square root term that arises from integrating over fluctuations around the classical solution (Occam factors). It was shown that Eq. (4) is nonsingular even at finite $N$, that the mean value of $\phi_{\text{cl}}$ converges to the negative logarithm of the target distribution $P(x)$ very quickly, and that the variance of fluctuations $\psi(x) \equiv \phi(x) - [-\log\ell_0 P(x)]$ falls off as $\sim 1/\sqrt{\ell N P(x)}$. Finally, it was speculated that if the actual $\ell$ is unknown one may average over it and hope that, much as in Bayesian model selection [2], the competition between the data and the fluctuations will select the optimal smoothness scale $\ell^*$.

At the first glance the theory seems to look almost exactly like a Gaussian Process [4]. This impression is produced by a Gaussian form of the smoothness penalty in Eq. (3), and by the fluctuation determinant that plays against the goodness of fit in the smoothness scale (model) selection. However, both similarities are incomplete. The Gaussian penalty in the prior is amended by the normalization constraint, which gives rise to the exponential term in Eq. (6), and violates many familiar results that hold for Gaussian Processes, the

representer theorem [12] being just one of them. In the semi–classical limit of large $N$, Gaussianity is restored approximately, but the classical solution is extremely non–trivial, and the fluctuation determinant is only the leading term of the Occam's razor, not the complete razor as it is for a Gaussian Process. In addition, it has no data dependence and is thus remarkably different from the usual determinants arising in the literature.

The algorithm to implement the discussed density estimation procedure numerically is rather simple. First, to make the problem well posed [10, 11] we confine $x$ to a box $0 \le x \le L$ with periodic boundary conditions. The boundary value problem Eq. (6) is then solved by a standard 'relaxation' (or Newton) method of iterative improvements to a guessed solution [13] (the target precision is always $10^{-5}$). The independent variable $x \in [0, 1]$ is discretized in equal steps [$10^4$ for Figs. (1.a–2.b), and $10^5$ for Figs. (3.a, 3.b)]. We use an equally spaced grid to ensure stability of the method, while small step sizes are needed since the scale for variation of $\phi_{cl}(x)$ is [5]

$$\delta x \sim \sqrt{\ell/NP(x)}\,, \tag{7}$$

which can be rather small for large $N$ or small $\ell$.

Since the theory is short scale insensitive, we can generate random probability densities chosen from the prior by replacing $\phi$ with its Fourier series and truncating the latter at some sufficiently high wavenumber $k_c$ [$k_c = 1000$ for Figs. (1.a–2.b), and 5000 for Figs. (3.a, 3.b)]. Then Eq. (3) enforces the amplitude of the $k$'th mode to be distributed a priori normally with the standard deviation

$$\sigma_k = \frac{2^{1/2}}{\ell^{\eta-1/2}} \left(\frac{L}{2\pi k}\right)^{\eta}. \tag{8}$$

Coded in such a way, the simulations are extremely computationally intensive. Therefore, Monte Carlo averagings given here are only over 500 runs, fluctuation determinants are calculated according to Eq. (5), not using numerical path integration, and $Q_{cl} = (1/\ell_0) \exp[-\phi_{cl}]$ is always used as an approximation to $Q_{est}$.

As an example of the algorithm's performance, Fig. (1.a) shows one particular learning run for $\eta = 1$ and $\ell = 0.2$. We see that singularities and overfitting are absent even for $N$ as low as 10. Moreover, the approach of $Q_{cl}(x)$ to the actual distribution $P(x)$ is remarkably fast: for $N = 10$, they are similar; for $N = 1000$, very close; for $N = 100000$, one needs to look carefully to see the difference between the two.

To quantify this similarity of distributions, we compute the Kullback–Leibler divergence $D_{KL}(P||Q_{est})$ between the true distribution $P(x)$ and its estimate $Q_{est}(x)$, and then average over the realizations of the data points and the true distribution. As discussed in [11], this learning curve $\Lambda(N)$ measures the (average) excess cost incurred in coding the $N+1$'st data point because of the finiteness of the data sample, and thus can be called the "universal learning curve". If the inference algorithm uses all of the information contained in the data that is relevant for learning ("predictive information" [11]), then [5, 9, 11, 10]

$$\Lambda(N) \sim (L/\ell)^{1/2\eta} N^{1/2\eta-1}. \tag{9}$$

We test this prediction against the learning curves in the actual simulations. For $\eta = 1$ and $\ell = 0.4, 0.2, 0.05$, these are shown on Fig. (1.b). One sees that the exponents are extremely close to the expected $1/2$, and the ratios of the prefactors are within the errors from the predicted scaling $\sim 1/\sqrt{\ell}$. All of this means that the proposed algorithm for finding densities not only works, but is at most a constant factor away from being optimal in using the predictive information of the sample set.

Next we investigate how one's choice of the prior influences learning. We first stress that there is no such thing as a *wrong prior*. If one admits a possibility of it being wrong, then

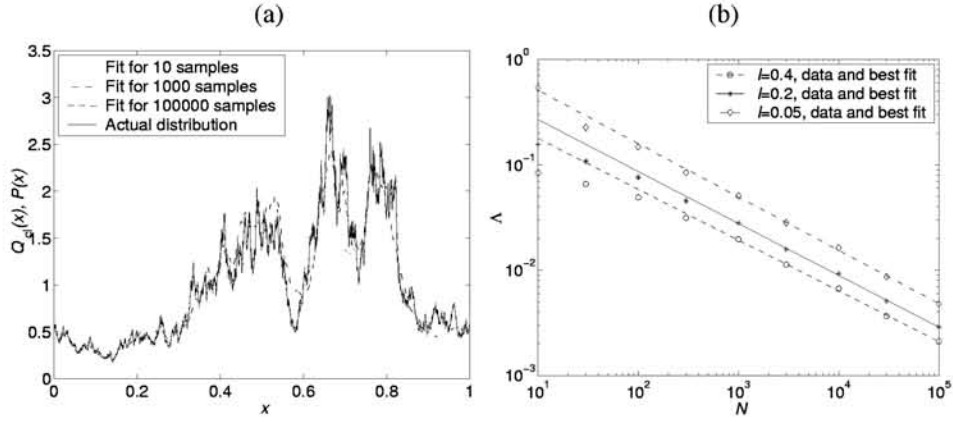

Figure 1: (a) $Q_{\mathrm{cl}}$ found for different $N$ at $\ell = 0.2$. (b) $\Lambda$ as a function of $N$ and $\ell$. The best fits are: for $\ell = 0.4$, $\Lambda = (0.54 \pm 0.07)N^{-0.483 \pm 0.014}$; for $\ell = 0.2$, $\Lambda = (0.83 \pm 0.08)N^{-0.493 \pm 0.09}$; for $\ell = 0.05$, $\Lambda = (1.64 \pm 0.16)N^{-0.507 \pm 0.09}$.

it does not encode all of the a priori knowledge! It does make sense, however, to ask what happens if the distribution we are trying to learn is an extreme outlier in the prior $\mathcal{P}[\phi]$. One way to generate such an example is to choose a typical function from a different prior $\mathcal{P}'[\phi]$, and this is what we mean by 'learning with a wrong prior.' If the prior is wrong in this sense, and learning is described by Eqs. (2–6), then we still expect the asymptotic behavior, Eq. (9), to hold; only the prefactors of $\Lambda$ should change, and those must increase since there is an obvious advantage in having the right prior; we illustrate this in Figs. (2.a, 2.b).

For Fig. (2.a), both $\mathcal{P}'[\phi]$ and $\mathcal{P}[\phi]$ are given by Eq. (3), but $\mathcal{P}'$ has the 'actual' smoothness scale $\ell_a = 0.4, 0.05$, and for $\mathcal{P}$ the 'learning' smoothness scale is $\ell = 0.2$ (we show the case $\ell_a = \ell = 0.2$ again as a reference). The $\Lambda \sim 1/\sqrt{N}$ behavior is seen unmistakably. The prefactors are a bit larger (unfortunately, insignificantly) than the corresponding ones from Fig. (1.b), so we may expect that the 'right' $\ell$, indeed, provides better learning (see later for a detailed discussion).

Further, Fig. (2.b) illustrates learning when not only $\ell$, but also $\eta$ is 'wrong' in the sense defined above. We illustrate this for $\eta_a = 2, 0.8, 0.6, 0$ (remember that only $\eta_a > 0.5$ removes UV divergences). Again, the inverse square root decay of $\Lambda$ should be observed, and this is evident for $\eta_a = 2$. The $\eta_a = 0.8, 0.6, 0$ cases are different: even for $N$ as high as $10^5$ the estimate of the distribution is far from the target, thus the asymptotic regime is not reached. This is a crucial observation for our subsequent analysis of the smoothness scale determination from the data. Remarkably, $\Lambda$ (both averaged and in the single runs shown) is monotonic, so even in the cases of *qualitatively* less smooth distributions *there still is no overfitting*. On the other hand, $\Lambda$ is well above the asymptote for $\eta = 2$ and small $N$, which means that initially too many details are expected and wrongfully introduced into the estimate, but then they are almost immediately ($N \sim 300$) eliminated by the data.

Following the argument suggested in [5], we now view $\mathcal{P}[\phi]$, Eq. (3), as being a part of some wider model that involves a prior over $\ell$. The details of the prior are irrelevant, however, if $S_{\mathrm{eff}}(\ell)$, Eq. (5), has a minimum that becomes more prominent as $N$ grows. We explicitly note that this mechanism is *not* tuning of the prior's parameters, but Bayesian inference at work: $\ell^*$ emerges in a competition between the smoothness, the data, and the Occam terms to make $S_{\mathrm{eff}}$ smaller, and thus the *total* probability of the data is larger. In its

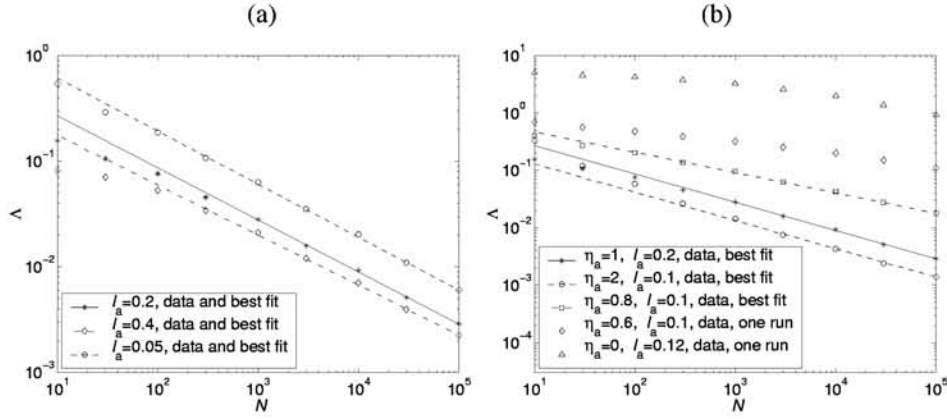

Figure 2: (a) $\Lambda$ as a function of $N$ and $\ell_a$. Best fits are: for $\ell_a = 0.4$, $\Lambda = (0.56 \pm 0.08)N^{-0.477\pm0.015}$; for $\ell_a = 0.05$, $\Lambda = (1.90 \pm 0.16)N^{-0.502\pm0.008}$. Learning is always with $\ell = 0.2$. (b) $\Lambda$ as a function of $N$, $\eta_a$ and $\ell_a$. Best fits: for $\eta_a = 2$, $\ell_a = 0.1$, $\Lambda = (0.40\pm0.05)N^{-0.493\pm0.013}$; for $\eta_a = 0.8$, $\ell_a = 0.1$, $\Lambda = (1.06\pm0.08)N^{-0.355\pm0.008}$. $\ell = 0.2$ for all graphs, but the one with $\eta_a = 0$, for which $\ell = 0.1$.

turn, larger probability means shorter total code length.

The data term, on average, is equal to $ND_{\mathrm{KL}}(P\|Q_{\mathrm{cl}})$, and, for very regular $P(x)$ (an implicit assumption in [5]), it is small. Thus only the kinetic and the Occam terms matter, and $\ell^* \sim N^{1/3}$[5]. For less regular distributions $P(x)$, this is not true [cf. Fig. (2.b)]. For $\eta = 1$, $Q_{\mathrm{cl}}(x)$ approximates large-scale features of $P(x)$ very well, but details at scales smaller than $\sim \sqrt{\ell/NL}$ are averaged out. If $P(x)$ is taken from the prior, Eq. (3), with some $\eta_a$, then these details fall off with the wave number $k$ as $\sim k^{-\eta_a}$. Thus the data term is $\sim N^{1.5-\eta_a}\ell^{\eta_a-0.5}$ and is not necessarily small. For $\eta_a < 1.5$ this dominates the kinetic term and competes with the fluctuations to set

$$\ell^* \sim N^{(\eta_a-1)/\eta_a}, \quad \eta_a < 1.5\,. \tag{10}$$

There are two remarkable things about Eq. (10). First, for $\eta_a = 1$, $\ell^*$ stabilizes at some constant value, which we expect to be equal to $\ell_a$. Second, even for $\eta \neq \eta_a$, Eqs. (9, 10) ensure that $\Lambda$ scales as $\sim N^{1/2\eta_a-1}$, which is at worst a constant factor away from the best scaling, Eq. (9), achievable with the 'right' prior, $\eta = \eta_a$. So, by allowing $\ell^*$ to vary with $N$ we can correctly capture the structure of models that are qualitatively different from our expectations ($\eta \neq \eta_a$) and produce estimates of $Q$ that are extremely robust to the choice of the prior. To our knowledge, this feature has not been noted before in a reference to a nonparametric problem.

We present simulations relevant to these predictions in Figs. (3.a, 3.b). Unlike on the previous Figures, the results are not averaged due to extreme computational costs, so all our further claims have to be taken cautiously. On the other hand, selecting $\ell^*$ in single runs has some practical advantages: we are able to ensure the best possible learning for any realization of the data. Fig. (3.a) shows single learning runs for various $\eta_a$ and $\ell_a$. In addition, to keep the Figure readable, we do not show runs with $\eta_a = 0.6, 0.7, 1.2, 1.5, 3$, and $\eta_a \to \infty$, which is a finitely parameterizable distribution. All of these display a good agreement with the predicted scalings: Eq. (10) for $\eta_a < 1.5$, and $\ell^* \sim N^{1/3}$ otherwise. Next we calculate the KL divergence between the target and the estimate at $\ell = \ell^*$; the average of this divergence over the samples and the prior is the learning curve [cf. Eq. (9)]. For $\eta_a = 0.8, 2$ we plot the divergencies on Fig. (3.b) side by side with their fixed $\ell = 0.2$

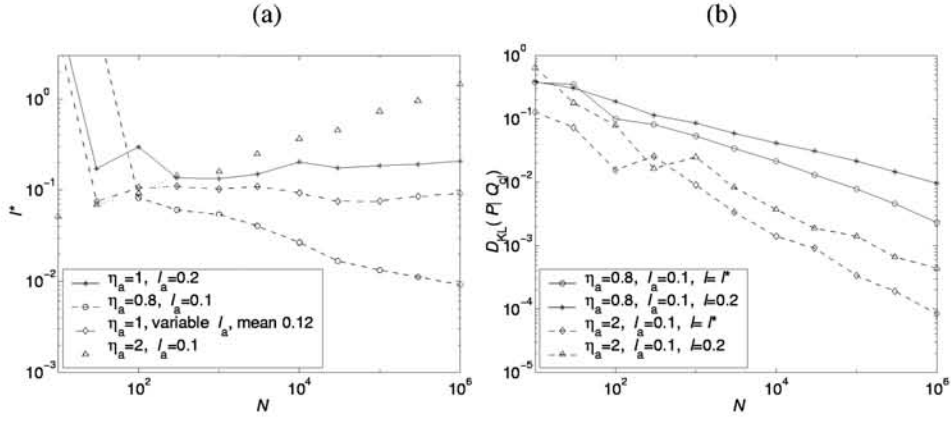

Figure 3: (a) Comparison of learning speed for the same data sets with different a priori assumptions. (b) Smoothness scale selection by the data. The lines that go off the axis for small $N$ symbolize that $S_{\text{eff}}$ monotonically decreases as $\ell \to \infty$.

analogues. Again, the predictions clearly are fulfilled. Note, that for $\eta_a \neq \eta$ there is a *qualitative* advantage in using the data induced smoothness scale.

The last four Figures have illustrated some aspects of learning with 'wrong' priors. However, all of our results may be considered as belonging to the 'wrong prior' class. Indeed, the actual probability distributions we used were not nonparametric continuous functions with smoothness constraints, but were composed of $k_c$ Fourier modes, thus had $2k_c$ parameters. For finite parameterization, asymptotic properties of learning usually do not depend on the priors (cf. [3, 11]), and priorless theories can be considered [14]. In such theories it would take well over $2k_c$ samples to even start to close down on the actual value of the parameters, and yet a lot more to get accurate results. However, using the wrong continuous parameterization [$\phi(x)$] we were able to obtain good fits for as low as 1000 samples [cf. Fig. (1.a)] with the help of the prior Eq. (3). Moreover, learning happened continuously and monotonically without huge chaotic jumps of overfitting that necessarily accompany any brute force parameter estimation method at low $N$. So, for some cases, a *seemingly more complex model* is actually *easier* to learn!

Thus our claim: when data are scarce and the parameters are abundant, one gains even by using the regularizing powers of wrong priors. The priors select some large scale features that are the most important to learn first and fill in the details as more data become available (see [11] on relation of this to the Structural Risk Minimization theory). If the global features are dominant (arguably, this is generic), one actually wins in the learning speed [cf. Figs. (1.b, 2.a, 3.b)]. If, however, small scale details are as important, then one at least is guaranteed to avoid overfitting [cf. Fig. (2.b)].

One can summarize this in an Occam-like fashion [11]: if two models provide equally good fits to data, *a simpler one should always be used*. In particular, the predictive information, which quantifies complexity [11], and of which $\Lambda$ is the derivative, in a QFT model is $\sim N^{1/2\eta}$, and it is $\sim k_c \log N$ in the parametric case. So, for $k_c > N^{1/2\eta}$, one should prefer a 'wrong' QFT formulation to the correct finite parameter model. These results are very much in the spirit of our whole program: not only is the value of $\ell^*$ selected that simplifies the description of the data, but the continuous parameterization itself serves the same purpose. This is an unexpectedly neat generalization of the MDL principle [3] to nonparametric cases.

*Summary:* The field theoretic approach to density estimation not only regularizes the learning process but also allows the self-consistent selection of smoothness criteria through an infinite dimensional version of the Occam factors. We have shown numerically that this works, even more clearly than was conjectured: for $\eta_a < 1.5$, the learning curve truly becomes a property of the data, and not of the Bayesian prior! If we can extend these results to other $\eta_a$ and combine this work with the reparameterization invariant formulation of [7, 8], this should give a complete theory of Bayesian learning for one dimensional distributions, and this theory has no arbitrary parameters. In addition, if this theory properly treats the limit $\eta_a \to \infty$, we should be able to see how the well–studied finite dimensional Occam factors and the MDL principle arise from a more general nonparametric formulation.

# References

[1] D. MacKay, *Neural Comp.* **4**, 415–448 (1992).

[2] V. Balasubramanian, *Neural Comp.* **9**, 349–368 (1997),
http://xxx.lanl.gov/abs/adap-org/9601001.

[3] J. Rissanen. *Stochastic Complexity and Statistical Inquiry*. World Scientific, Singapore (1989).

[4] D. MacKay, NIPS, Tutorial Lecture Notes (1997),
ftp://wol.ra.phy.cam.ac.uk/pub/mackay/gp.ps.gz.

[5] W. Bialek, C. Callan, and S. Strong, *Phys. Rev. Lett.* **77**, 4693–4697 (1996),
http://xxx.lanl.gov/abs/cond-mat/9607180.

[6] T. Holy, *Phys. Rev. Lett.* **79**, 3545–3548 (1997),
http://xxx.lanl.gov/abs/physics/9706015.

[7] V. Periwal, *Phys. Rev. Lett.* **78**, 4671–4674 (1997),
http://xxx.lanl.gov/hep-th/9703135.

[8] V. Periwal, *Nucl. Phys. B*, **554** [FS], 719-730 (1999),
http://xxx.lanl.gov/adap-org/9801001.

[9] T. Aida, *Phys. Rev. Lett.* **83**, 3554–3557 (1999),
http://xxx.lanl.gov/cond-mat/9911474.

[10] A more detailed version of our current analysis may be found in: I. Nemenman, Ph.D. Thesis, Princeton, (2000), http://xxx.lanl.gov/abs/physics/0009032.

[11] W. Bialek, I. Nemenman, N. Tishby. Preprint
http://xxx.lanl.gov/abs/physics/0007070.

[12] G. Wahba. In B. Shölkopf, C. J. S. Burges, and A. J. Smola, eds., *Advances in Kernel Methods—Support Vector Learning*, pp. 69-88. MIT Press, Cambridge, MA (1999),
ftp://ftp.stat.wisc.edu/pub/wahba/nips97rr.ps.

[13] W. Press et al. *Numerical Recipes in C*. Cambridge UP, Cambridge (1988).

[14] Vapnik, V. *Statistical Learning Theory*. John Wiley & Sons, New York (1998).